# ADAPTIVE NEURAL NET PREPROCESSING FOR SIGNAL DETECTION IN NON-GAUSSIAN NOISE[1]

Richard P. Lippmann and Paul Beckman
MIT Lincoln Laboratory
Lexington, MA 02173

## ABSTRACT

A nonlinearity is required before matched filtering in minimum error receivers when additive noise is present which is impulsive and highly non-Gaussian. Experiments were performed to determine whether the correct clipping nonlinearity could be provided by a single-input single-output multi-layer perceptron trained with back propagation. It was found that a multi-layer perceptron with one input and output node, 20 nodes in the first hidden layer, and 5 nodes in the second hidden layer could be trained to provide a clipping nonlinearity with fewer than 5,000 presentations of noiseless and corrupted waveform samples. A network trained at a relatively high signal-to-noise (S/N) ratio and then used as a front end for a linear matched filter detector greatly reduced the probability of error. The clipping nonlinearity formed by this network was similar to that used in current receivers designed for impulsive noise and provided similar substantial improvements in performance.

## INTRODUCTION

The most widely used neural net, the adaptive linear combiner (ALC), is a single-layer perceptron with linear input and output nodes. It is typically trained using the LMS algorithm and forms one of the most common components of adaptive filters. ALCs are used in high-speed modems to construct equalization filters, in telephone links as echo cancelers, and in many other signal processing applications where linear filtering is required [9]. The purpose of this study was to determine whether multi-layer perceptrons with linear input and output nodes but with sigmoidal hidden nodes could be as effective for adaptive nonlinear filtering as ALCs are for linear filtering.

The task explored in this paper is signal detection with impulsive noise where an adaptive nonlinearity is required for optimal performance. Impulsive noise occurs in underwater acoustics and in extremely low frequency communications channels where impulses caused by lightning strikes propagate many thousands of miles [2]. This task was selected because a nonlinearity is required in the optimal receiver, the structure of the optimal receiver is known, and the resulting signal detection error rate provides an objective measure of performance. The only other previous studies of the use of multi-layer perceptrons for adaptive nonlinear filtering that we are aware of [6,8] appear promising but provide no objective performance comparisons.

In the following we first present examples which illustrate that multi-layer perceptrons trained with back-propagation can rapidly form clipping and other nonlinearities useful for signal processing with deterministic training. The signal detection task is then described and theory is presented which illustrates the need for non-linear processing with non-Gaussian noise. Nonlinearities formed when the input to a net is a corrupted signal and the desired output is the uncorrupted signal are then presented for no noise, impulsive noise, and Gaussian noise. Finally, signal detection performance results are presented that demonstrate large improvements in performance with an adaptive nonlinearity and impulsive noise.

## FORMING DETERMINISTIC NONLINEARITIES

A theorem proven by Kolmogorov and described in [5] demonstrates that single-input single-output continuous nonlinearities can be formed by a multi-layer perceptron with two layers of hidden nodes. This proof, however, requires complex nonlinear functions in the hidden nodes that are very sensitive to the desired input/output function and may be difficult to realize. More recently, Lapedes [4] presented an intuitive description of how multi-layer perceptrons with sigmoidal nonlinearities could produce continuous nonlinear mappings. A careful mathematical proof was recently developed by Cybenko [1] which demonstrated that continuous nonlinear mappings can be formed using sigmoidal nonlinearities and a multi-layer perceptron with one layer of hidden nodes. This proof, however, is not constructive and does not indicate how many nodes are required in the hidden layer. The purpose of our study was to determine whether multi-layer perceptrons with sigmoidal nonlinearities and trained using back-propagation could adaptively and rapidly form clipping nonlinearities.

Initial experiments were performed to determine the difficulty of learning complex mappings using multi-layer perceptrons trained using back-propagation. Networks with 1 and 2 hidden layers and from 1 to 50 hidden nodes per layer were evaluated. Input and output nodes were linear and all other nodes included sigmoidal nonlinearities. Best overall performance was provided by the three-layer perceptron shown in Fig. 1. It has 20 nodes in the first and 5 nodes in the second hidden layer. This network could form a wide variety of mappings and required only slightly more training than other networks. It was used in all experiments.

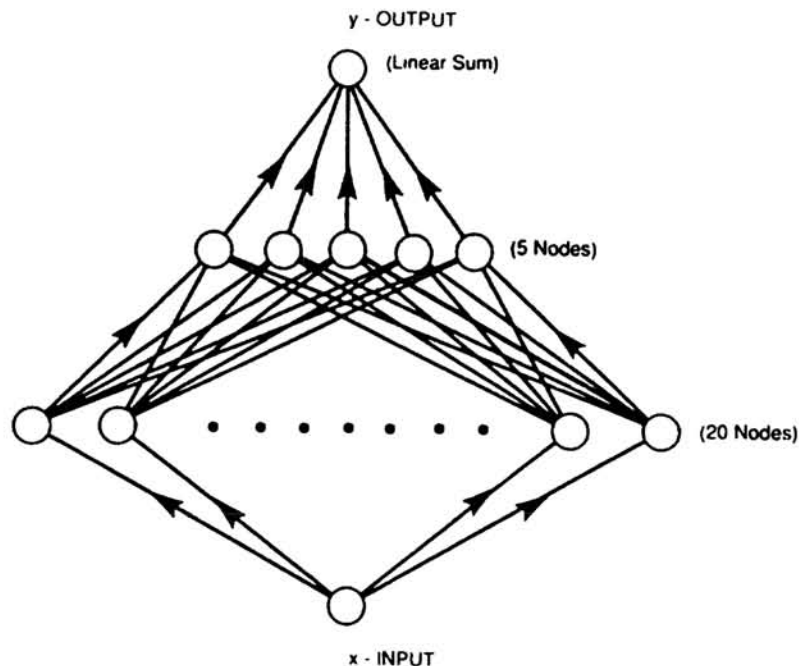

Figure 1: The multi-layer perceptron with linear input and output nodes that was used in all experiments.

The three-layer network shown in Fig. 1 was used to form clipping and other deterministic nonlinearities. Results in Fig. 2 demonstrate that a clipping nonlinearity could be formed with fewer than 1,000 input samples. Input/output point pairs were determined by selecting the input at random over the range plotted and using the deterministic clipping function shown as a solid line in Fig. 2. Back-propagation training [7] was used with the gain term ($\eta$) equal to 0.1 and the momentum term ($\alpha$) equal to 0.5. These values provide good convergence rates for the clipping function and all other functions tested. Initial connection weights were set to small random values.

The multi-layer perceptron from Fig. 1 was also used to form the four nonlinear functions shown in Fig. 3. The "Hole Punch" is useful in nonlinear signal processing. It performs much the same function as the clipper but completely eliminates amplitudes above a certain threshold level. Accurate approximation of this function required more than 50,000 input samples. The "Step" has one sharp edge and could be roughly approximated after 2,000 input samples. The "Double Pulse" requires approximation of two close "pulses" and is the nonlinear function analogy of the disjoint region problem studied in [3]. In this example, back-propagation training approximated the rightmost pulse first after 5,000 input samples. Both pulses were then approximated fairly well after 50,000 input samples. The "Gaussian Pulse" is a smooth curve that could be approximated well after only 2,000 input samples. These results demonstrate that back-propagation training with sigmoidal nonlinearities can form many different nonlinear functions. Qualitative results on training times are similar to those reported in [3]. In this previous study it was demon-

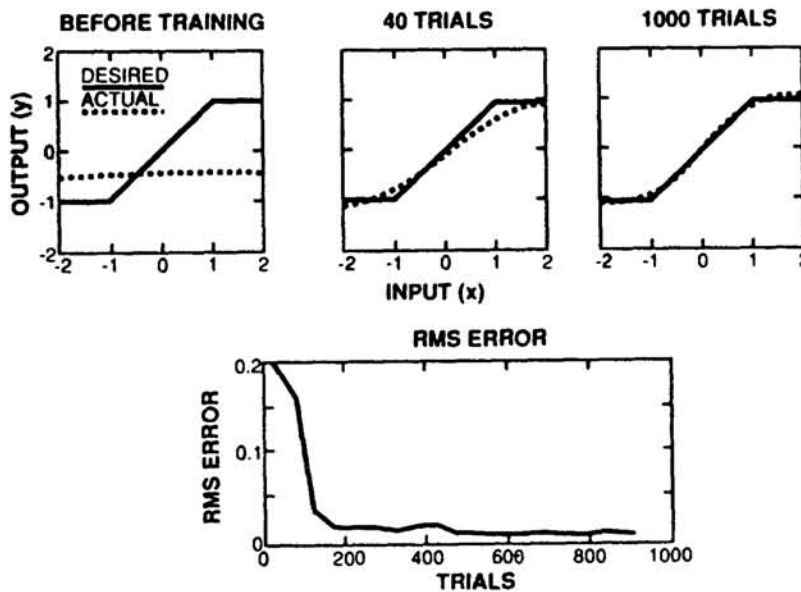

Figure 2: Clipping nonlinearities formed using back-propagation training and the multi-layer perceptron from Fig. 1 (top) and the rms error produced by these nonlinearities versus training time (bottom).

strated that simple half-plane decision regions could be formed for classification problems with little training while complex disjoint decision regions required long training times. These new results suggest that complex nonlinearities with many sharp discontinuities require much more training time than simple smooth curves.

## THE SIGNAL DETECTION TASK

The signal detection task was to discriminate between two equally likely input signals as shown in Fig. 4. One signal ($s_0(t)$) corresponds to no input and the other signal ($s_1(t)$) was a sinewave pulse with fixed duration and known amplitude, frequency, and phase. Noise was added to these inputs, the resultant signal was passed through a memoryless nonlinearity, and a matched filter was then used to select hypothesis $H_0$ corresponding to no input or $H_1$ corresponding to the sinewave pulse. The matched filter multiplied the output of the nonlinearity by the known time-aligned signal waveform, integrated this product over time, and decided $H_1$ if the result was greater than a threshold and $H_0$ otherwise. The threshold was selected to provide a minimum overall error rate. The optimum nonlinearity used in the detector depends on the noise distribution. If the signal levels are small relative to the noise levels, then the optimum nonlinearity is approximated by $f(x) = \frac{d}{dx} \ln(p_n(x))$, where $p_n(x)$ is the instantaneous probability density function of the noise [2]. This function is linear for Gaussian noise but has a clipping shape for impulsive noise.

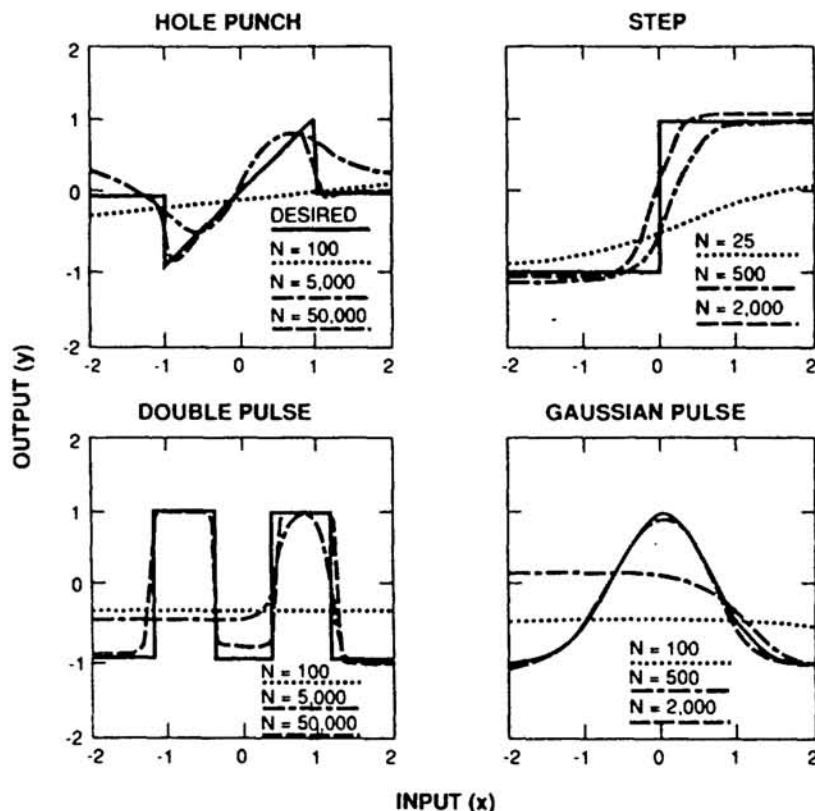

Figure 3: Four deterministic nonlinearities formed using the multi-layer perceptron from Fig. 1. Desired functions are plotted as solid lines while functions formed using back-propagation with different numbers of input samples are plotted using dots and dashes.

Examples of the signal, impulsive noise and Gaussian noise are presented in Fig. 5. The signal had a fixed duration of 250 samples and peak amplitude of 1.0. The impulsive noise was defined by its amplitude distribution and inter-arrival time. Amplitudes had a zero mean, Laplacian distribution with a standard deviation ($\sigma$) of 14.1 in all experiments. The standard deviation was reduced to 2.8 in Fig. 5 for illustrative purposes. Inter-arrival times ($\Delta T$) between noise impulses had a Poisson distribution. The mean inter-arrival time was varied in experiments to obtain different S/N ratios after adding noise. For example varying inter-arrival times from 500 to 2 samples results in S/N ratios that vary from roughly 1 $dB$ to $-24$ $dB$. Additive Gaussian noise had zero mean and a standard deviation ($\sigma$) of 0.1 in all experiments.

## ADAPTIVE TRAINING WITH NOISE

The three-layer perceptron was trained as shown in Fig. 6 using the signal plus noise as the input and the uncorrupted signal as the desired output. Network weights were adapted after every sample input using back-propagation training. Adaptive nonlinearities formed during training are shown in Fig. 7. These are similar to those

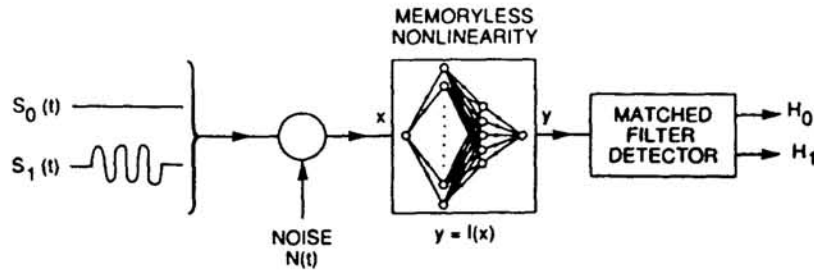

Figure 4: The signal detection task was to discriminate between a sinewave pulse and a no-input condition with additive impulsive noise.

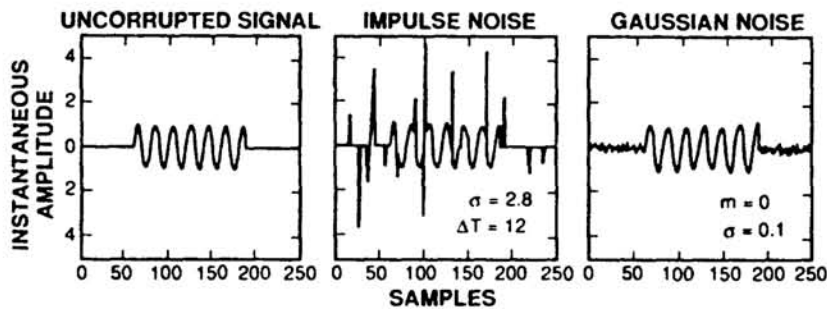

Figure 5: The input to the nonlinearity with no noise, additive impulsive noise, and additive Gaussian noise.

required by theory. No noise results in nonlinearity that is linear over the range of the input sinewave ($-1$ to $+1$) after fewer than 3,000 input samples. Impulsive noise at a high S/N ratio ($\Delta T = 125$ or $S/N = -5$ $dB$) results in a nonlinearity that clips above the signal level after roughly 5,000 input samples and then slowly forms a "Hole Punch" nonlinearity as the number of training samples increases. Gaussian noise noise results in a nonlinearity that is roughly linear over the range of the input sinewave after fewer than 5,000 input samples.

## SIGNAL DETECTION PERFORMANCE

Signal detection performance was measured using a matched filter detector and the nonlinearity shown in the center of Fig. 7 for 10,000 input training samples. The error rate with a minimum-error matched filter is plotted in Fig. 8 for impulsive noise at S/N ratios ranging from roughly 5 $dB$ to $-24$ $dB$. This error rate was estimated from 2,000 signal detection trials. Signal detection performance always improved with the nonlinearity and sometimes the improvement was dramatic. The error rate provided with the adaptively-formed nonlinearity is essentially identical

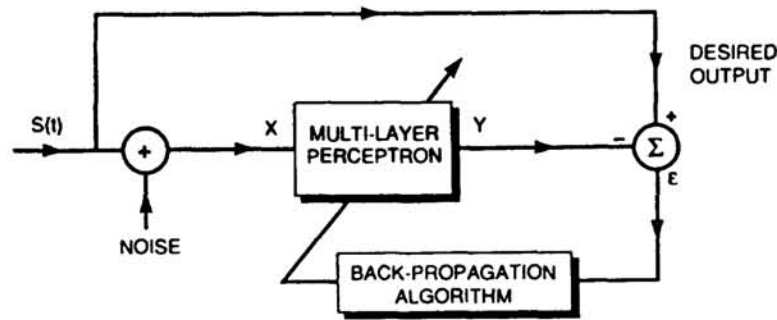

Figure 6: The procedure used for adaptive training.

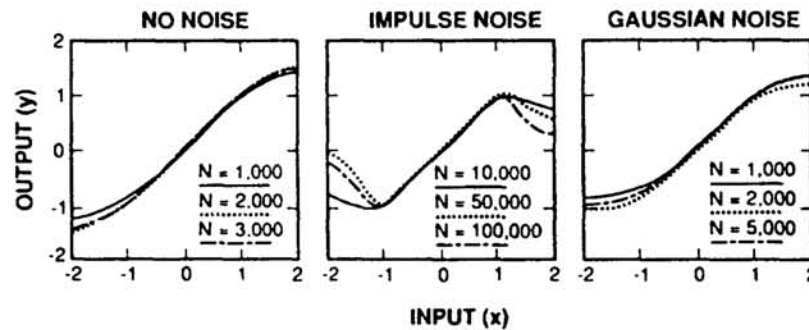

Figure 7: Nonlinearities formed with adaptive training with no additive noise, with additive impulsive noise at a S/N level of −5 $dB$, and with additive Gaussian noise.

to that provided by a clipping nonlinearity that clips above the signal level. This error rate is roughly zero down to −24 $dB$ and then rises rapidly with higher levels of impulsive noise. This rapid increase in error rate below −24 $dB$ is not shown in Fig. 8. The error rate with linear processing rises slowly as the S/N ratio drops and reaches roughly 36% when the S/N ratio is −24 $dB$.

Further exploratory experiments demonstrated that the nonlinearity formed by back-propagation was not robust to the S/N ratio used during training. A clipping nonlinearity is only formed when the number of samples of uncorrupted sinewave input is high enough to form the linear function of the curve and the number of samples of noise pulses is low, but sufficient to form the nonlinear clipping section of the nonlinearity. At high noise levels the resulting nonlinearity is not linear over the range of the input signal. It instead resembles a curve that interpolates between a flat horizontal input-output curve and the desired clipping curve.

## SUMMARY AND DISCUSSION

In summary, it was first demonstrated that multi-layer perceptrons with linear

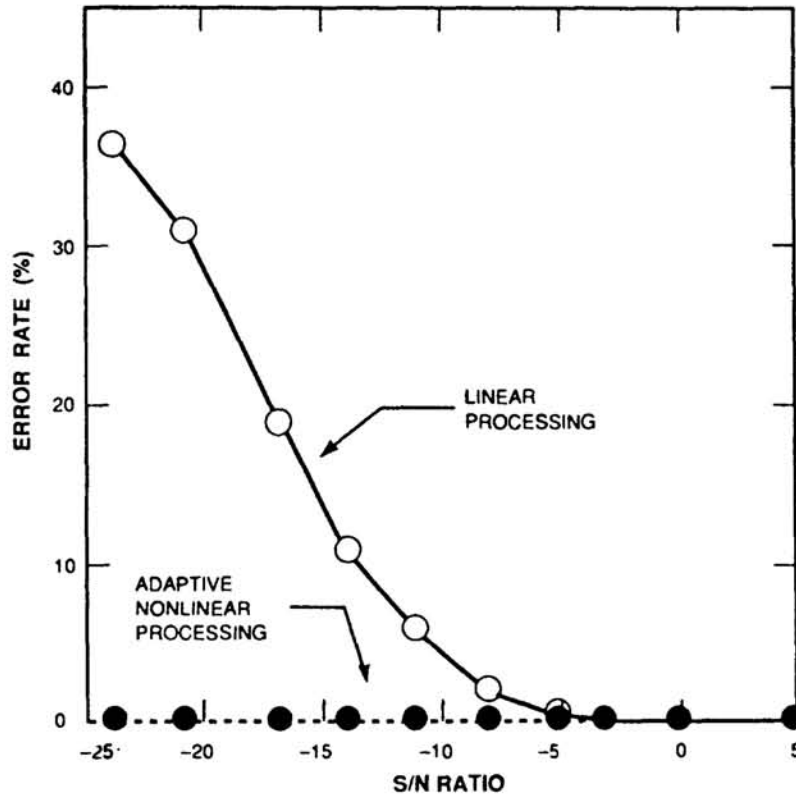

Figure 8: The signal detection error rate with impulsive noise when the S/N ratio after adding the noise ranges from 5 $dB$ to $-24$ $dB$.

input and output nodes could approximate prespecified clipping nonlinearities required for signal detection with impulsive noise with fewer than 1,000 trials of back-propagation training. More complex nonlinearities could also be formed but required longer training times. Clipping nonlinearities were also formed adaptively using a multi-layer perceptron with the corrupted signal as the input and the noise-free signal as the desired output. Nonlinearities learned using this approach at high S/N ratios were similar to those required by theory and improved signal detection performance dramatically at low S/N ratios. Further work is necessary to further explore the utility of this technique for forming adaptive nonlinearities. This work should explore the robustness of the nonlinearity formed to variations in the input S/N ratio. It should also explore the use of multi-layer perceptrons and back-propagation training for other adaptive nonlinear signal processing tasks such as system identification, noise removal, and channel modeling.

## Footnotes

[1] This work was sponsored by the Defense Advanced Research Projects Agency and the Department of the Air Force. The views expressed are those of the authors and do not reflect the policy or position of the U. S. Government.

# References

[1] G. Cybenko. Approximation by superpositions of a sigmoidal function. Research note, Department of Computer Science, Tufts University, October 1988.

[2] J. E. Evans and A. S Griffiths. Design of a sanguine noise processor based upon world-wide extremely low frequency (elf) recordings. *IEEE Transactions on Communications*, COM-22:528–539, April 1974.

[3] W. M. Huang and R. P. Lippmann. Neural net and traditional classifiers. In D. Anderson, editor, *Neural Information Processing Systems*, pages 387–396, New York, 1988. American Institute of Physics.

[4] A. Lapedes and R. Farber. How neural nets work. In D. Anderson, editor, *Neural Information Processing Systems*, pages 442–456, New York, 1988. American Institute of Physics.

[5] G. G. Lorentz. The 13th problem of Hilbert. In F. E. Browder, editor, *Mathematical Developments Arising from Hilbert Problems*. American Mathematical Society, Providence, R.I., 1976.

[6] D. Palmer and D. DeSieno. Removing random noise from ekg signals using a back propagation network, 1987. HNC Inc., San Diego, CA.

[7] D. E. Rumelhart, G. E. Hinton, and R. J. Williams. Learning internal representations by error propagation. In D. E. Rumelhart and J. L. McClelland, editors, *Parallel Distributed Processing*, volume 1: Foundations, chapter 8. MIT Press, Cambridge, MA, 1986.

[8] S. Tamura and A. Wailbel. Noise reduction using connectionist models. In *Proceedings IEEE International Conference on Acoustics, Speech and Signal Processing*, volume 1: Speech Processing, pages 553–556, April 1988.

[9] B. Widrow and S. D. Stearns. *Adaptive Signal Processing*. Prentice-Hall, NJ, 1985.